# Fitted Q-iteration in continuous action-space MDPs

**András Antos**
Computer and Automation Research Inst.
of the Hungarian Academy of Sciences
Kende u. 13-17, Budapest 1111, Hungary
antos@sztaki.hu

**Rémi Munos**
SequeL project-team, INRIA Lille
59650 Villeneuve d'Ascq, France
remi.munos@inria.fr

**Csaba Szepesvári**[*]
Department of Computing Science
University of Alberta
Edmonton T6G 2E8, Canada
szepesva@cs.ualberta.ca

## Abstract

We consider continuous state, continuous action batch reinforcement learning where the goal is to learn a good policy from a sufficiently rich trajectory generated by some policy. We study a variant of fitted Q-iteration, where the greedy action selection is replaced by searching for a policy in a restricted set of candidate policies by maximizing the average action values. We provide a rigorous analysis of this algorithm, proving what we believe is the first finite-time bound for value-function based algorithms for continuous state and action problems.

## 1 Preliminaries

We will build on the results from [1, 2, 3] and for this reason we use the same notation as these papers. The unattributed results cited in this section can be found in the book [4].

A discounted MDP is defined by a quintuple $(\mathcal{X}, \mathcal{A}, P, S, \gamma)$, where $\mathcal{X}$ is the (possible infinite) *state space*, $\mathcal{A}$ is the set of *actions*, $P : \mathcal{X} \times \mathcal{A} \to M(\mathcal{X})$ is the *transition probability kernel* with $P(\cdot|x, a)$ defining the next-state distribution upon taking action $a$ from state $x$, $S(\cdot|x, a)$ gives the corresponding distribution of *immediate rewards*, and $\gamma \in (0, 1)$ is the discount factor. Here $\mathcal{X}$ is a measurable space and $M(\mathcal{X})$ denotes the set of all probability measures over $\mathcal{X}$. The Lebesgue-measure shall be denoted by $\lambda$. We start with the following mild assumption on the MDP:

**Assumption A1 (MDP Regularity)** $\mathcal{X}$ is a compact subset of the $d_{\mathcal{X}}$-dimensional Euclidean space, $\mathcal{A}$ is a compact subset of $[-A_{\infty}, A_{\infty}]^{d_{\mathcal{A}}}$. The random immediate rewards are bounded by $\hat{R}_{\max}$ and that the expected immediate reward function, $r(x, a) = \int r S(dr|x, a)$, is uniformly bounded by $R_{\max}$: $\|r\|_{\infty} \le R_{\max}$.

A *policy* determines the next action given the past observations. Here we shall deal with *stationary (Markovian)* policies which choose an action in a stochastic way based on the last observation only.

The *value* of a policy $\pi$ when it is started from a state $x$ is defined as the total expected discounted reward that is encountered while the policy is executed: $V^{\pi}(x) = \mathbb{E}_{\pi} \left[ \sum_{t=0}^{\infty} \gamma^t R_t | X_0 = x \right]$. Here $R_t \sim S(\cdot|X_t, A_t)$ is the reward received at time step $t$, the state, $X_t$, evolves according to $X_{t+1} \sim$

---

[*]Also with: Computer and Automation Research Inst. of the Hungarian Academy of Sciences Kende u. 13-17, Budapest 1111, Hungary.

$P(\cdot|X_t, A_t)$, where $A_t$ is sampled from the distribution determined by $\pi$. We use $Q^\pi : \mathcal{X} \times \mathcal{A} \to \mathbb{R}$ to denote the *action-value function* of policy $\pi$: $Q^\pi(x,a) = \mathbb{E}_\pi \left[ \sum_{t=0}^\infty \gamma^t R_t | X_0 = x, A_0 = a \right]$.

The goal is to find a policy that attains the best possible values, $V^*(x) = \sup_\pi V^\pi(x)$, at all states $x \in \mathcal{X}$. Here $V^*$ is called the *optimal value function* and a policy $\pi^*$ that satisfies $V^{\pi^*}(x) = V^*(x)$ for all $x \in \mathcal{X}$ is called *optimal*. The *optimal action-value function* $Q^*(x,a)$ is $Q^*(x,a) = \sup_\pi Q^\pi(x,a)$. We say that a (deterministic stationary) policy $\pi$ is *greedy* w.r.t. an action-value function $Q \in B(\mathcal{X} \times \mathcal{A})$, and we write $\pi = \hat{\pi}(\cdot; Q)$, if, for all $x \in \mathcal{X}$, $\pi(x) \in \operatorname{argmax}_{a \in \mathcal{A}} Q(x,a)$. Under mild technical assumptions, such a greedy policy always exists. Any greedy policy w.r.t. $Q^*$ is optimal. For $\pi : \mathcal{X} \to \mathcal{A}$ we define its *evaluation operator*, $T^\pi : B(\mathcal{X} \times \mathcal{A}) \to B(\mathcal{X} \times \mathcal{A})$, by $(T^\pi Q)(x,a) = r(x,a) + \gamma \int_{\mathcal{X}} Q(y, \pi(y)) \, P(dy|x,a)$. It is known that $Q^\pi = T^\pi Q^\pi$. Further, if we let the *Bellman operator*, $T : B(\mathcal{X} \times \mathcal{A}) \to B(\mathcal{X} \times \mathcal{A})$, defined by $(TQ)(x,a) = r(x,a) + \gamma \int_{\mathcal{X}} \sup_{b \in \mathcal{A}} Q(y,b) \, P(dy|x,a)$ then $Q^* = TQ^*$. It is known that $V^\pi$ and $Q^\pi$ are bounded by $R_{\max}/(1 - \gamma)$, just like $Q^*$ and $V^*$. For $\pi : \mathcal{X} \to \mathcal{A}$, the operator $E^\pi : B(\mathcal{X} \times \mathcal{A}) \to B(\mathcal{X})$ is defined by $(E^\pi Q)(x) = Q(x, \pi(x))$, while $E : B(\mathcal{X} \times \mathcal{A}) \to B(\mathcal{X})$ is defined by $(EQ)(x) = \sup_{a \in \mathcal{A}} Q(x,a)$.

Throughout the paper $\mathcal{F} \subset \{f : \mathcal{X} \times \mathcal{A} \to \mathbb{R}\}$ will denote a subset of real-valued functions over the state-action space $\mathcal{X} \times \mathcal{A}$ and $\Pi \subset \mathcal{A}^{\mathcal{X}}$ will be a set of policies. For $\nu \in M(\mathcal{X})$ and $f : \mathcal{X} \to \mathbb{R}$ measurable, we let (for $p \geq 1$) $\|f\|_{p,\nu}^p = \int_{\mathcal{X}} |f(x)|^p \nu(dx)$. We simply write $\|f\|_\nu$ for $\|f\|_{2,\nu}$. Further, we extend $\|\cdot\|_\nu$ to $\mathcal{F}$ by $\|f\|_\nu^2 = \int_{\mathcal{A}} \int_{\mathcal{X}} |f|^2(x,a) \, d\nu(x) \, d\lambda_{\mathcal{A}}(a)$, where $\lambda_{\mathcal{A}}$ is the uniform distribution over $\mathcal{A}$. We shall use the shorthand notation $\nu f$ to denote the integral $\int f(x) \nu(dx)$. We denote the space of bounded measurable functions with domain $\mathcal{X}$ by $B(\mathcal{X})$. Further, the space of measurable functions bounded by $0 < K < \infty$ shall be denoted by $B(\mathcal{X}; K)$. We let $\|\cdot\|_\infty$ denote the supremum norm.

## 2  Fitted Q-iteration with approximate policy maximization

We assume that we are given a finite trajectory, $\{(X_t, A_t, R_t)\}_{1 \leq t \leq N}$, generated by some stochastic stationary policy $\pi_b$, called the *behavior policy*: $A_t \sim \pi_b(\cdot|X_t)$, $X_{t+1} \sim P(\cdot|X_t, A_t)$, $R_t \sim S(\cdot|X_t, A_t)$, where $\pi_b(\cdot|x)$ is a density with $\pi_0 \overset{\text{def}}{=} \inf_{(x,a) \in \mathcal{X} \times \mathcal{A}} \pi_b(a|x) > 0$.

The generic recipe for fitted Q-iteration (FQI) [5] is

$$Q_{k+1} = \operatorname{Regress}(D_k(Q_k)), \tag{1}$$

where Regress is an appropriate regression procedure and $D_k(Q_k)$ is a dataset defining a regression problem in the form of a list of data-point pairs:

$$D_k(Q_k) = \left\{ \left[ (X_t, A_t), R_t + \gamma \max_{b \in \mathcal{A}} Q_k(X_{t+1}, b) \right]_{1 \leq t \leq N} \right\}.^1$$

Fitted Q-iteration can be viewed as approximate value iteration applied to action-value functions. To see this note that value iteration would assign the value $(TQ_k)(x,a) = r(x,a) + \gamma \int \max_{b \in \mathcal{A}} Q_k(y,b) \, P(dy|x,a)$ to $Q_{k+1}(x,a)$ [6]. Now, remember that the regression function for the jointly distributed random variables $(Z, Y)$ is defined by the conditional expectation of $Y$ given $Z$: $m(Z) = \mathbb{E}[Y|Z]$. Since for any *fixed* function $Q$, $\mathbb{E}[R_t + \gamma \max_{b \in \mathcal{A}} Q(X_{t+1}, b)|X_t, A_t] = (TQ)(X_t, A_t)$, the regression function corresponding to the data $D_k(Q)$ is indeed $TQ$ and hence if FQI solved the regression problem defined by $Q_k$ exactly, it would simulate value iteration exactly.

However, this argument itself does not directly lead to a rigorous analysis of FQI: Since $Q_k$ is obtained based on the data, it is itself a random function. Hence, after the first iteration, the "target" function in FQI becomes random. Furthermore, this function depends on the same data that is used to define the regression problem. Will FQI still work despite these issues? To illustrate the potential difficulties consider a dataset where $X_1, \ldots, X_N$ is a sequence of independent random variables, which are all distributed uniformly at random in $[0,1]$. Further, let $M$ be a random integer greater than $N$ which is independent of the dataset $(X_t)_{t=1}^N$. Let $U$ be another random variable, uniformly distributed in $[0,1]$. Now define the regression problem by $Y_t = f_{M,U}(X_t)$, where $f_{M,U}(x) = \operatorname{sgn}(\sin(2^M 2\pi(x + U)))$. Then it is not hard to see that no matter how big $N$ is, no procedure can

estimate the regression function $f_{M,U}$ with a small error (in expectation, or with high probability), even if the procedure could exploit the knowledge of the specific form of $f_{M,U}$. On the other hand, if we restricted $M$ to a finite range then the estimation problem could be solved successfully. The example shows that if the complexity of the random functions defining the regression problem is uncontrolled then successful estimation might be impossible.

Amongst the many regression methods in this paper we have chosen to work with least-squares methods. In this case Equation (1) takes the form

$$Q_{k+1} \;\; = \;\; \underset{Q \in \mathcal{F}}{\operatorname{argmin}} \sum_{t=1}^{N} \frac{1}{\pi_b(A_t|X_t)} \left( Q(X_t, A_t) - \left[ R_t + \gamma \max_{b \in \mathcal{A}} Q_k(X_{t+1}, b) \right] \right)^2. \qquad (2)$$

We call this method the least-squares fitted Q-iteration (LSFQI) method. Here we introduced the weighting $1/\pi_b(A_t|X_t)$ since we do not want to give more weight to those actions that are preferred by the behavior policy.

Besides this weighting, the only parameter of the method is the function set $\mathcal{F}$. This function set should be chosen carefully, to keep a balance between the representation power and the number of samples. As a specific example for $\mathcal{F}$ consider neural networks with some fixed architecture. In this case the function set is generated by assigning weights in all possible ways to the neural net. Then the above minimization becomes the problem of tuning the weights. Another example is to use linearly parameterized function approximation methods with appropriately selected basis functions. In this case the weight tuning problem would be less demanding. Yet another possibility is to let $\mathcal{F}$ be an appropriate restriction of a Reproducing Kernel Hilbert Space (e.g., in a ball). In this case the training procedure becomes similar to LS-SVM training [7].

As indicated above, the analysis of this algorithm is complicated by the fact that the new dataset is defined in terms of the previous iterate, which is already a function of the dataset. Another complication is that the samples in a trajectory are in general correlated and that the bias introduced by the imperfections of the approximation architecture may yield to an explosion of the error of the procedure, as documented in a number of cases in, e.g., [8].

Nevertheless, at least for finite action sets, the tools developed in [1, 3, 2] look suitable to show that under appropriate conditions these problems can be overcome if the function set is chosen in a judicious way. However, the results of these works would become essentially useless in the case of an infinite number of actions since these previous bounds grow to infinity with the number of actions. Actually, we believe that this is not an artifact of the proof techniques of these works, as suggested by the counterexample that involved random targets. The following result elaborates this point further:

**Proposition 2.1.** *Let $\mathcal{F} \subset B(\mathcal{X} \times \mathcal{A})$. Then even if the pseudo-dimension of $\mathcal{F}$ is finite, the fat-shattering function of*

$$\mathcal{F}_{\max}^{\vee} = \left\{ V_Q \; : \; V_Q(\cdot) = \max_{a \in \mathcal{A}} Q(\cdot, a), Q \in \mathcal{F} \right\}$$

*can be infinite over $(0, 1/2)$.[2]*

Without going into further details, let us just note that the finiteness of the fat-shattering function is a sufficient and necessary condition for learnability and the finiteness of the fat-shattering function is implied by the finiteness of the pseudo-dimension [9].The above proposition thus shows that without imposing further special conditions on $\mathcal{F}$, the learning problem may become infeasible.

One possibility is of course to discretize the action space, e.g., by using a uniform grid. However, if the action space has a really high dimensionality, this approach becomes unfeasible (even enumerating $2^{d_{\mathcal{A}}}$ points could be impossible when $d_{\mathcal{A}}$ is large). Therefore we prefer alternate solutions.

Another possibility is to make the functions in $\mathcal{F}$, e.g., uniformly Lipschitz in their state coordinates. Then the same property will hold for functions in $\mathcal{F}_{\max}^{\vee}$ and hence by a classical result we can bound the capacity of this set (cf. pp. 353–357 of [10]). One potential problem with this approach is that this way it might be difficult to get a fine control of the capacity of the resulting set.

In the approach explored here we modify the fitted Q-iteration algorithm by introducing a policy set $\Pi$ and a search over this set for an approximately greedy policy in a sense that will be made precise in a minute. Our algorithm thus has four parameters: $\mathcal{F}, \Pi, K, Q_0$. Here $\mathcal{F}$ is as before, $\Pi$ is a user-chosen set of policies (mappings from $\mathcal{X}$ to $\mathcal{A}$), $K$ is the number of iterations and $Q_0$ is an initial value function (a typical choice is $Q_0 \equiv 0$). The algorithm computes a sequence of iterates $(Q_k, \hat{\pi}_k)$, $k = 0, \ldots, K$, defined by the following equations:

$$\hat{\pi}_0 = \underset{\pi \in \Pi}{\operatorname{argmax}} \sum_{t=1}^{N} Q_0(X_t, \pi(X_t)),$$

$$Q_{k+1} = \underset{Q \in \mathcal{F}}{\operatorname{argmin}} \sum_{t=1}^{N} \frac{1}{\pi_b(A_t|X_t)} \Big( Q(X_t, A_t) - \big[ R_t + \gamma Q_k(X_{t+1}, \hat{\pi}_k(X_{t+1})) \big] \Big)^2, \quad (3)$$

$$\hat{\pi}_{k+1} = \underset{\pi \in \Pi}{\operatorname{argmax}} \sum_{t=1}^{N} Q_{k+1}(X_t, \pi(X_t)). \quad (4)$$

Thus, (3) is similar to (2), while (4) defines the policy search problem. The policy search will generally be solved by a gradient procedure or some other appropriate method. The cost of this step will be primarily determined by how well-behaving the iterates $Q_{k+1}$ are in their action arguments. For example, if they were quadratic and if $\pi$ was linear then the problem would be a quadratic optimization problem. However, except for special cases[3] the action value functions will be more complicated, in which case this step can be expensive. Still, this cost could be similar to that of searching for the maximizing actions for each $t = 1, \ldots, N$ if the approximately maximizing actions are similar across similar states.

This algorithm, which we could also call a fitted *actor-critic* algorithm, will be shown to overcome the above mentioned complexity control problem provided that the complexity of $\Pi$ is controlled appropriately. Indeed, in this case the set of possible regression problems is determined by the set

$$\mathcal{F}_\Pi^\vee = \{ V \, : \, V(\cdot) = Q(\cdot, \pi(\cdot)), Q \in \mathcal{F}, \pi \in \Pi \},$$

and the proof will rely on controlling the complexity of $\mathcal{F}_\Pi^\vee$ by selecting $\mathcal{F}$ and $\Pi$ appropriately.

## 3 The main theoretical result

### 3.1 Outline of the analysis

In order to gain some insight into the behavior of the algorithm, we provide a brief summary of its error analysis. The main result will be presented subsequently. For $f, Q \in \mathcal{F}$ and a policy $\pi$, we define the $t^{\text{th}}$ TD-error as follows:

$$d_t(f; Q, \pi) = R_t + \gamma Q(X_{t+1}, \pi(X_{t+1})) - f(X_t, A_t).$$

Further, we define the empirical loss function by

$$\hat{L}_N(f; Q, \pi) = \frac{1}{N} \sum_{t=1}^{N} \frac{d_t^2(f; Q, \pi)}{\lambda(\mathcal{A})\pi_b(A_t|X_t)},$$

where the normalization with $\lambda(\mathcal{A})$ is introduced for mathematical convenience. Then (3) can be written compactly as $Q_{k+1} = \operatorname{argmin}_{f \in \mathcal{F}} \hat{L}_N(f; Q_k, \hat{\pi}_k)$.

The algorithm can then be motivated by the observation that for any $f, Q$, and $\pi$, $\hat{L}_N(f; Q, \pi)$ is an unbiased estimate of

$$L(f; Q, \pi) \stackrel{\text{def}}{=} \|f - T^\pi Q\|_\nu^2 + L^*(Q, \pi), \quad (5)$$

where the first term is the error we are interested in and the second term captures the variance of the random samples:

$$L^*(Q, \pi) = \int_{\mathcal{A}} \mathbb{E}\left[ \operatorname{Var}\left[ R_1 + \gamma Q(X_2, \pi(X_2)) | X_1, A_1 = a \right] \right] d\lambda_{\mathcal{A}}(a).$$

This result is stated formally by $\mathbb{E}\left[\hat{L}_N(f; Q, \pi)\right] = L(f; Q, \pi)$.

Since the variance term in (5) is independent of $f$, $\operatorname{argmin}_{f \in \mathcal{F}} L(f; Q, \pi) = \operatorname{argmin}_{f \in \mathcal{F}} \|f - T^\pi Q\|_\nu^2$. Thus, if $\hat{\pi}_k$ were greedy w.r.t. $Q_k$ then $\operatorname{argmin}_{f \in \mathcal{F}} L(f; Q_k, \hat{\pi}_k) = \operatorname{argmin}_{f \in \mathcal{F}} \|f - TQ_k\|_\nu^2$. Hence we can still think of the procedure as approximate value iteration over the space of action-value functions, projecting $TQ_k$ using empirical risk minimization on the space $\mathcal{F}$ w.r.t. $\|\cdot\|_\nu$ distances in an approximate manner. Since $\hat{\pi}_k$ is only approximately greedy, we will have to deal with both the error coming from the approximate projection and the error coming from the choice of $\hat{\pi}_k$. To make this clear, we write the iteration in the form

$$Q_{k+1} = T^{\hat{\pi}_k} Q_k + \varepsilon_k' = TQ_k + \varepsilon_k' + (T^{\hat{\pi}_k} Q_k - TQ_k) = TQ_k + \varepsilon_k,$$

where $\varepsilon_k'$ is the error committed while computing $T^{\hat{\pi}_k} Q_k$, $\varepsilon_k'' \stackrel{\text{def}}{=} T^{\hat{\pi}_k} Q_k - TQ_k$ is the error committed because the greedy policy is computed approximately and $\varepsilon_k = \varepsilon_k' + \varepsilon_k''$ is the total error of step $k$. Hence, in order to show that the procedure is well behaved, one needs to show that both errors are controlled and that when the errors are propagated through these equations, the resulting error stays controlled, too. Since we are ultimately interested in the performance of the policy obtained, we will also need to show that small action-value approximation errors yield small performance losses. For these we need a number of assumptions that concern either the training data, the MDP, or the function sets used for learning.

### 3.2 Assumptions

#### 3.2.1 Assumptions on the training data

We shall assume that the data is rich, is in a steady state, and is fast-mixing, where, informally, mixing means that future depends weakly on the past.

**Assumption A2 (Sample Path Properties)** Assume that $\{(X_t, A_t, R_t)\}_{t=1,\ldots,N}$ is the sample path of $\pi_b$, a stochastic stationary policy. Further, assume that $\{X_t\}$ is strictly stationary ($X_t \sim \nu \in M(\mathcal{X})$) and exponentially $\beta$-mixing with the actual rate given by the parameters $(\bar{\beta}, b, \kappa)$.[4] We further assume that the sampling policy $\pi_b$ satisfies $\pi_0 = \inf_{(x,a) \in \mathcal{X} \times \mathcal{A}} \pi_b(a|x) > 0$.

The $\beta$-mixing property will be used to establish tail inequalities for certain empirical processes.[5] Note that the mixing coefficients do not need to be known. In the case when no mixing condition is satisfied, learning might be impossible. To see this just consider the case when $X_1 = X_2 = \ldots = X_N$. Thus, in this case the learner has many copies of the same random variable and successful generalization is thus impossible. We believe that the assumption that the process is in a steady state is not essential for our result, as when the process reaches its steady state quickly then (at the price of a more involved proof) the result would still hold.

#### 3.2.2 Assumptions on the MDP

In order to prevent the uncontrolled growth of the errors as they are propagated through the updates, we shall need some assumptions on the MDP. A convenient assumption is the following one [11]:

**Assumption A3 (Uniformly stochastic transitions)** For all $x \in \mathcal{X}$ and $a \in \mathcal{A}$, assume that $P(\cdot|x, a)$ is absolutely continuous w.r.t. $\nu$ and the Radon-Nikodym derivative of $P$ w.r.t. $\nu$ is bounded uniformly with bound $C_\nu$: $C_\nu \stackrel{\text{def}}{=} \sup_{x \in \mathcal{X}, a \in \mathcal{A}} \left\|\frac{dP(\cdot|x,a)}{d\nu}\right\|_\infty < +\infty$.

Note that by the definition of measure differentiation, Assumption A3 means that $P(\cdot|x, a) \leq C_\nu \nu(\cdot)$. This assumption essentially requires the transitions to be noisy. We will also prove (weaker) results under the following, *weaker* assumption:

**Assumption A4 (Discounted-average concentrability of future-state distributions)** Given $\rho$, $\nu$, $m \geq 1$ and an arbitrary sequence of stationary policies $\{\pi_m\}_{m\geq 1}$, assume that the future-state distribution $\rho P^{\pi_1} P^{\pi_2} \ldots P^{\pi_m}$ is absolutely continuous w.r.t. $\nu$. Assume that $c(m) \stackrel{\text{def}}{=} \sup_{\pi_1,\ldots,\pi_m} \left\| \frac{d(\rho P^{\pi_1} P^{\pi_2} \ldots P^{\pi_m})}{d\nu} \right\|_\infty$ satisfies $\sum_{m\geq 1} m\gamma^{m-1} c(m) < +\infty$. We shall call $C_{\rho,\nu} \stackrel{\text{def}}{=} \max\left\{ (1-\gamma)^2 \sum_{m\geq 1} m\gamma^{m-1} c(m), (1-\gamma) \sum_{m\geq 1} \gamma^m c(m) \right\}$ the *discounted-average concentrability coefficient* of the future-state distributions.

The number $c(m)$ measures how much $\rho$ can get amplified in $m$ steps as compared to the reference distribution $\nu$. Hence, in general we expect $c(m)$ to grow with $m$. In fact, the condition that $C_{\rho,\mu}$ is finite is a growth rate condition on $c(m)$. Thanks to discounting, $C_{\rho,\mu}$ is finite for a reasonably large class of systems (see the discussion in [11]).

A related assumption is needed in the error analysis of the approximate greedy step of the algorithm:

**Assumption A5 (The random policy "makes no peak-states")** Consider the distribution $\mu = (\nu \times \lambda_{\mathcal{A}})P$ which is the distribution of a state that results from sampling an initial state according to $\nu$ and then executing an action which is selected uniformly at random.[6] Then $\Gamma_\nu = \|d\mu/d\nu\|_\infty < +\infty$.

Note that under Assumption A3 we have $\Gamma_\nu \leq C_\nu$. This (very mild) assumption means that after one step, starting from $\nu$ and executing this random policy, the probability of the next state being in a set is upper bounded by $\Gamma_\nu$-times the probability of the starting state being in the same set.

Besides, we assume that $\mathcal{A}$ has the following regularity property: Let $\mathrm{Py}(a,h,\rho) \stackrel{\text{def}}{=} \left\{ (a',v) \in \mathbb{R}^{d_{\mathcal{A}}+1} : \|a - a'\|_1 \leq \rho, 0 \leq v/h \leq 1 - \|a - a'\|_1/\rho \right\}$ denote the pyramid with hight $h$ and base given by the $\ell^1$-ball $B(a,\rho) \stackrel{\text{def}}{=} \left\{ a' \in \mathbb{R}^{d_{\mathcal{A}}} : \|a - a'\|_1 \leq \rho \right\}$ centered at $a$.

**Assumption A6 (Regularity of the action space)** We assume that there exists $\alpha > 0$, such that for all $a \in \mathcal{A}$, for all $\rho > 0$,

$$\frac{\lambda(\mathrm{Py}(a,1,\rho) \cap (\mathcal{A} \times \mathbb{R}))}{\lambda(\mathrm{Py}(a,1,\rho))} \geq \min\left(\alpha, \frac{\lambda(\mathcal{A})}{\lambda(B(a,\rho))}\right).$$

For example, if $\mathcal{A}$ is an $\ell^1$-ball itself, then this assumption will be satisfied with $\alpha = 2^{-d_{\mathcal{A}}}$.

Without assuming any smoothness of the MDP, learning in *infinite* MDPs looks hard (see, e.g., [12, 13]). Here we employ the following extra condition:

**Assumption A7 (Lipschitzness of the MDP in the actions)** Assume that the transition probabilities and rewards are Lipschitz w.r.t. their action variable, i.e., there exists $L_P, L_r > 0$ such that for all $(x,a,a') \in \mathcal{X} \times \mathcal{A} \times \mathcal{A}$ and measurable set $B$ of $\mathcal{X}$,

$$|P(B|x,a) - P(B|x,a')| \leq L_P \|a - a'\|_1, \quad |r(x,a) - r(x,a')| \leq L_r \|a - a'\|_1.$$

Note that previously Lipschitzness w.r.t. the *state* variables was used, e.g., in [11] to construct consistent planning algorithms.

### 3.2.3 Assumptions on the function sets used by the algorithm

These assumptions are less demanding since they are under the control of the user of the algorithm. However, the choice of these function sets will greatly influence the performance of the algorithm, as we shall see it from the bounds. The first assumption concerns the class $\mathcal{F}$:

**Assumption A8 (Lipschitzness of candidate action-value functions)** Assume $\mathcal{F} \subset B(\mathcal{X} \times \mathcal{A})$ and that any elements of $\mathcal{F}$ is uniformly Lipschitz in its action-argument in the sense that $|Q(x,a) - Q(x,a')| \leq L_{\mathcal{A}} \|a - a'\|_1$ holds for any $x \in \mathcal{X}$, $a, a' \in \mathcal{A}$, and $Q \in \mathcal{F}$.

We shall also need to control the capacity of our function sets. We assume that the reader is familiar with the concept of VC-dimension.[7] Here we use the *pseudo-dimension* of function sets that builds upon the concept of VC-dimension:

**Definition 3.1** (Pseudo-dimension). *The* pseudo-dimension $V_{\mathcal{F}^+}$ *of* $\mathcal{F}$ *is defined as the* VC-*dimension of the subgraphs of functions in* $\mathcal{F}$ *(hence it is also called the* VC-subgraph dimension *of* $\mathcal{F}$*).*

Since $\mathcal{A}$ is multidimensional, we define $V_{\Pi^+}$ to be the sum of the pseudo-dimensions of the coordinate projection spaces, $\Pi_k$ of $\Pi$:

$$V_{\Pi^+} = \sum_{k=1}^{d_{\mathcal{A}}} V_{\Pi_k^+}, \quad \Pi_k = \left\{ \pi_k : \mathcal{X} \to \mathbb{R} : \pi = (\pi_1, \ldots, \pi_k, \ldots, \pi_{d_{\mathcal{A}}}) \in \Pi \right\}.$$

Now we are ready to state our assumptions on our function sets:

**Assumption A9 (Capacity of the function and policy sets)** Assume that $\mathcal{F} \subset B(\mathcal{X} \times \mathcal{A}; Q_{\max})$ for $Q_{\max} > 0$ and $V_{\mathcal{F}^+} < +\infty$. Also, $\mathcal{A} \subset [-A_\infty, A_\infty]^{d_{\mathcal{A}}}$ and $V_{\Pi^+} < +\infty$.

Besides their capacity, one shall also control the approximation power of the function sets involved. Let us first consider the policy set $\Pi$. Introduce

$$e^*(\mathcal{F}, \Pi) = \sup_{Q \in \mathcal{F}} \inf_{\pi \in \Pi} \nu(EQ - E^\pi Q).$$

Note that $\inf_{\pi \in \Pi} \nu(EQ - E^\pi Q)$ measures the quality of approximating $\nu EQ$ by $\nu E^\pi Q$. Hence, $e^*(\mathcal{F}, \Pi)$ measures the worst-case approximation error of $\nu EQ$ as $Q$ is changed within $\mathcal{F}$. This can be made small by choosing $\Pi$ large.

Another related quantity is the *one-step Bellman-error of* $\mathcal{F}$ *w.r.t.* $\Pi$. This is defined as follows: For a fixed policy $\pi$, the one-step Bellman-error of $\mathcal{F}$ w.r.t. $T^\pi$ is defined as

$$E_1(\mathcal{F}; \pi) = \sup_{Q \in \mathcal{F}} \inf_{Q' \in \mathcal{F}} \|Q' - T^\pi Q\|_\nu.$$

Taking again a pessimistic approach, the one-step Bellman-error of $\mathcal{F}$ is defined as

$$E_1(\mathcal{F}, \Pi) = \sup_{\pi \in \Pi} E_1(\mathcal{F}; \pi).$$

Typically by increasing $\mathcal{F}$, $E_1(\mathcal{F}, \Pi)$ can be made smaller (this is discussed at some length in [3]). However, it also holds for both $\Pi$ and $\mathcal{F}$ that making them bigger will increase their capacity (pseudo-dimensions) which leads to an increase of the estimation errors. Hence, $\mathcal{F}$ and $\Pi$ must be selected to balance the approximation and estimation errors, just like in supervised learning.

### 3.3 The main result

**Theorem 3.2.** *Let* $\pi_K$ *be a greedy policy w.r.t.* $Q_K$, *i.e.* $\pi_K(x) \in \arg\max_{a \in \mathcal{A}} Q_K(x, a)$. *Then under Assumptions A1, A2, and A5–A9, for all* $\delta > 0$ *we have with probability at least* $1 - \delta$*: given Assumption A3 (respectively A4),* $\|V^* - V^{\pi_K}\|_\infty$ *(resp.* $\|V^* - V^{\pi_K}\|_{1,\rho}$*), is bounded by*

$$C \left\{ \left( E_1(\mathcal{F}, \Pi) + e^*(\mathcal{F}, \Pi) + \frac{(\log N + \log(K/\delta))^{\frac{\kappa+1}{4\kappa}}}{N^{1/4}} \right)^{\frac{1}{d_{\mathcal{A}}+1}} + \gamma^K \right\},$$

*where* $C$ *depends on* $d_{\mathcal{A}}$, $V_{\mathcal{F}^+}$, $(V_{\Pi_k^+})_{k=1}^{d_{\mathcal{A}}}$, $\gamma$, $\kappa$, $b$, $\overline{\beta}$, $C_\nu$ *(resp.* $C_{\rho,\nu}$*),* $\Gamma_\nu$, $L_{\mathcal{A}}$, $L_P$, $L_r$, $\alpha$, $\lambda(\mathcal{A})$, $\pi_0$, $Q_{\max}$, $R_{\max}$, $\hat{R}_{\max}$, *and* $A_\infty$. *In particular,* $C$ *scales with* $V^{\frac{\kappa+1}{4\kappa(d_{\mathcal{A}}+1)}}$, *where* $V = 2V_{\mathcal{F}^+} + V_{\Pi^+}$ *plays the role of the "combined effective" dimension of* $\mathcal{F}$ *and* $\Pi$.

# 4 Discussion

We have presented what we believe is the first finite-time bounds for continuous-state and action-space RL that uses value functions. Further, this is the first analysis of fitted Q-iteration, an algorithm that has proved to be useful in a number of cases, even when used with non-averagers for which no previous theoretical analysis existed (e.g., [15, 16]). In fact, our main motivation was to show that there is a systematic way of making these algorithms work and to point at possible problem sources the same time. We discussed why it can be difficult to make these algorithms work in practice. We suggested that either the set of action-value candidates has to be carefully controlled (e.g., assuming uniform Lipschitzness w.r.t. the state variables), or a policy search step is needed, just like in actor-critic algorithms. The bound in this paper is similar in many respects to a previous bound of a Bellman-residual minimization algorithm [2]. It looks that the techniques developed here can be used to obtain results for that algorithm when it is applied to continuous action spaces. Finally, although we have not explored them here, consistency results for FQI can be obtained from our results using standard methods, like the methods of sieves. We believe that the methods developed here will eventually lead to algorithms where the function approximation methods are chosen based on the data (similar to adaptive regression methods) so as to optimize performance, which in our opinion is one of the biggest open questions in RL. Currently we are exploring this possibility.

## Acknowledgments

András Antos would like to acknowledge support for this project from the Hungarian Academy of Sciences (Bolyai Fellowship). Csaba Szepesvári greatly acknowledges the support received from the Alberta Ingenuity Fund, NSERC, the Computer and Automation Research Institute of the Hungarian Academy of Sciences.

## Footnotes

[1] Since the designer controls $Q_k$, we may assume that it is continuous, hence the maximum exists.

[2]The proof of this and the other results are given in the appendix, available in the extended version of this paper, downloadable from http://hal.inria.fr/inria-00185311/en/.

[3]Linear quadratic regulation is such a nice case. It is interesting to note that in this special case the obvious choices for $\mathcal{F}$ and $\Pi$ yield zero error in the limit, as can be proven based on the main result of this paper.

[4]For the definition of $\beta$-mixing, see e.g. [2].

[5]We say "empirical process" and "empirical measure", but note that in this work these are based on dependent (mixing) samples.

[6]Remember that $\lambda_{\mathcal{A}}$ denotes the uniform distribution over the action set $\mathcal{A}$.

[7]Readers not familiar with VC-dimension are suggested to consult a book, such as the one by Anthony and Bartlett [14].

# References

[1] A. Antos, Cs. Szepesvári, and R. Munos. Learning near-optimal policies with Bellman-residual minimization based fitted policy iteration and a single sample path. In *COLT-19*, pages 574–588, 2006.

[2] A. Antos, Cs. Szepesvári, and R. Munos. Learning near-optimal policies with Bellman-residual minimization based fitted policy iteration and a single sample path. *Machine Learning*, 2007. (accepted).

[3] A. Antos, Cs. Szepesvári, and R. Munos. Value-iteration based fitted policy iteration: learning with a single trajectory. In *IEEE ADPRL*, pages 330–337, 2007.

[4] D. P. Bertsekas and S.E. Shreve. *Stochastic Optimal Control (The Discrete Time Case)*. Academic Press, New York, 1978.

[5] D. Ernst, P. Geurts, and L. Wehenkel. Tree-based batch mode reinforcement learning. *Journal of Machine Learning Research*, 6:503–556, 2005.

[6] R.S. Sutton and A.G. Barto. *Reinforcement Learning: An Introduction*. Bradford Book. MIT Press, 1998.

[7] N. Cristianini and J. Shawe-Taylor. *An introduction to support vector machines (and other kernel-based learning methods)*. Cambridge University Press, 2000.

[8] J.A. Boyan and A.W. Moore. Generalization in reinforcement learning: Safely approximating the value function. In *NIPS-7*, pages 369–376, 1995.

[9] P.L. Bartlett, P.M. Long, and R.C. Williamson. Fat-shattering and the learnability of real-valued functions. *Journal of Computer and System Sciences*, 52:434–452, 1996.

[10] A.N. Kolmogorov and V.M. Tihomirov. $\epsilon$-entropy and $\epsilon$-capacity of sets in functional space. *American Mathematical Society Translations*, 17(2):277–364, 1961.

[11] R. Munos and Cs. Szepesvári. Finite time bounds for sampling based fitted value iteration. Technical report, Computer and Automation Research Institute of the Hungarian Academy of Sciences, Kende u. 13-17, Budapest 1111, Hungary, 2006.

[12] A.Y. Ng and M. Jordan. PEGASUS: A policy search method for large MDPs and POMDPs. In *Proceedings of the 16th Conference in Uncertainty in Artificial Intelligence*, pages 406–415, 2000.

[13] P.L. Bartlett and A. Tewari. Sample complexity of policy search with known dynamics. In *NIPS-19*. MIT Press, 2007.

[14] M. Anthony and P. L. Bartlett. *Neural Network Learning: Theoretical Foundations*. Cambridge University Press, 1999.

[15] M. Riedmiller. Neural fitted Q iteration – first experiences with a data efficient neural reinforcement learning method. In *16th European Conference on Machine Learning*, pages 317–328, 2005.

[16] S. Kalyanakrishnan and P. Stone. Batch reinforcement learning in a complex domain. In *AAMAS-07*, 2007.

